# Planning with an Adaptive World Model

**Sebastian B. Thrun**
German National Research
Center for Computer
Science (GMD)
D-5205 St. Augustin, FRG

**Knut Möller**
University of Bonn
Department of
Computer Science
D-5300 Bonn, FRG

**Alexander Linden**
German National Research
Center for Computer
Science (GMD)
D-5205 St. Augustin, FRG

## Abstract

We present a new connectionist planning method [TML90]. By interaction with an unknown environment, a world model is progressively constructed using gradient descent. For deriving optimal actions with respect to future reinforcement, planning is applied in two steps: an experience network proposes a plan which is subsequently optimized by gradient descent with a chain of world models, so that an optimal reinforcement may be obtained when it is actually run. The appropriateness of this method is demonstrated by a robotics application and a pole balancing task.

## 1  INTRODUCTION

Whenever decisions are to be made with respect to some events in the future, planning has been proved to be an important and powerful concept in problem solving. Planning is applicable if an autonomous agent interacts with a world, and if a reinforcement is available which measures only the over-all performance of the agent. Then the problem of optimizing actions yields the *temporal credit assignment problem* [Sut84], i.e. the problem of assigning particular reinforcements to particular actions in the past. The problem becomes more complicated if no knowledge about the world is available in advance.

Many connectionist approaches so far solve this problem directly, using techniques based on the interaction of an adaptive world model and an adaptive controller [Bar89, Jor89, Mun87]. Although such controllers are very fast after training, training itself is rather complex, mainly because of two reasons: a) Since future is not considered explicitly, future effects must be directly encoded into the world model. This complicates model training. b) Since the controller is trained with the world model, training of the former lags behind the latter. Moreover, if there do exist

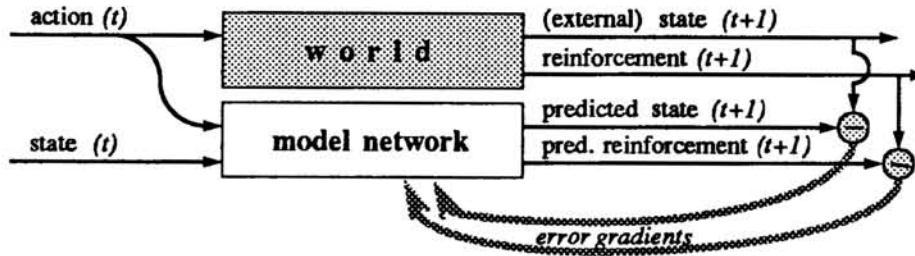

Figure 1: The training of the model network is a system identification task. Internal parameters are estimated by gradient descent, e.g. by backpropagation.

several optimal actions, such controllers will only generate at most one regardless of all others, since they represent many-to-one functions. E.g., changing the objective function implies the need of an expensive retraining.

In order to overcome these problems, we applied a planning technique to reinforcement learning problems. A model network which approximates the behavior of the world is used for looking ahead into future and optimizing actions by gradient descent with respect to future reinforcement. In addition, an experience network is trained in order to accelerate and improve planning.

## 2 LOOK-AHEAD PLANNING

### 2.1 SYSTEM IDENTIFICATION

Planning needs a world model. Training of the world model is adopted from [Bar89, Jor89, Mun87]. Formally, the world maps actions to subsequent states and reinforcements (Fig. 1). The world model used here is a standard non-recurrent or a recurrent connectionist network which is trained by backpropagation or related gradient descent algorithms [WZ88, TS90]. Each time an action is performed on the world their resulting state and reinforcement is compared with the corresponding prediction by the model network. The difference is used for adapting the internal parameters of the model in small steps, in order to improve its accuracy. The resulting model approximates the world's behavior.

Our planning technique relies mainly on two fundamental steps: Firstly, a plan is proposed either by some heuristic or by a so-called *experience network*. Secondly, this plan is optimized progressively by gradient descent in action space. First, we will consider the second step.

### 2.2 PLAN OPTIMIZATION

In this section we show the optimization of plans by means of gradient descent. For that purpose, let us assume an initial plan, i.e. a sequence of $N$ actions, is given. The first action of this plan together with the current state (and, in case of a recurrent model network, its current context activations) are fed into the model network (Fig. 2). This gives us a prediction for the subsequent state and reinforcement of the world. If we assume that the state prediction is a good estimation for the next state, we can proceed by predicting the immediate next state and reinforcement from the second action of the plan correspondingly. This procedure is repeated for each of the $N$ stages of the plan. The final output is a sequence of $N$ reinforcement predictions, which represents the quality of the plan. In order to maximize reinforcement, we

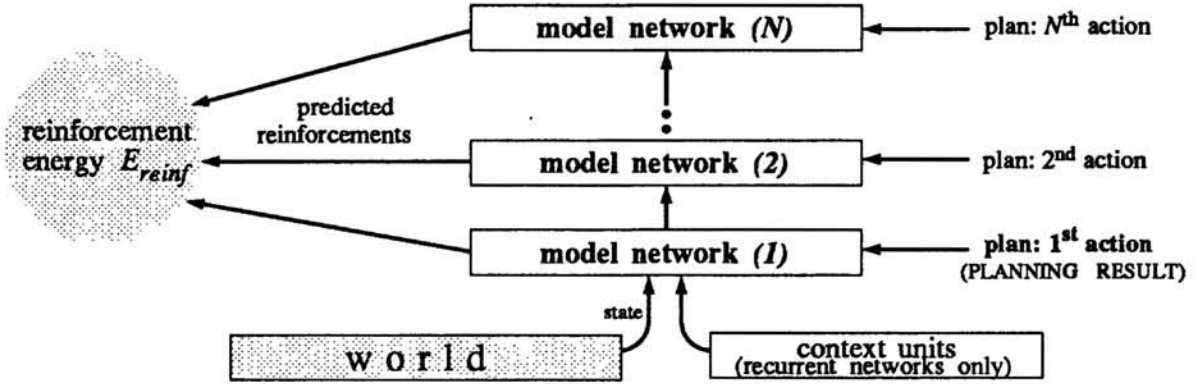

Figure 2: Looking ahead by the chain of model networks.

establish a differentiable *reinforcement energy* function $E_{\text{reinf}}$, which measures the deviation of predicted and desired reinforcement. The problem of optimizing plans is transformed to the problem of minimizing $E_{\text{reinf}}$. Since both $E_{\text{reinf}}$ and the chain of model networks are differentiable, the gradients of the plan with respect to $E_{\text{reinf}}$ can be computed. These gradients are used for changing the plan in small steps, which completes the gradient descent optimization.

The whole update procedure is repeated either until convergence is observed or, which makes it more convenient for real-time applications, a predefined number of iterations – note that in the latter case the computational effort is linear in $N$. From the planning procedure we obtain the optimized plan, the first action[1] of which is then performed on the world. Now the whole procedure is repeated.

The gradients of the plan with respect to $E_{\text{reinf}}$ can be computed either by back-propagation through the chain of models or by a feed-forward algorithm which is related to [WZ88, TS90]:
Hand in hand with the activations we propagate also the gradients

$$\xi_{is}^j\,(\tau) \;\;\equiv\;\; \frac{\partial \text{ activation}_j\,(\tau)}{\partial \text{ action}_i\,(s)} \tag{1}$$

through the chain of models. Here $i$ labels all action input units and $j$ all units of the whole model network, $\tau\ (1\leq\tau\leq N)$ is the time associated with the $\tau$th model of the chain, and $s\ (1\leq s\leq\tau)$ is the time of the $s$th action. Thus, for each action $(\forall i, s)$ its influence on later activations $(\forall j, \forall\tau\geq s)$ of the chain of networks, including all predictions, is measured by $\xi_{is}^j(\tau)$.

It has been shown in an earlier paper that this gradient can easily be propagated forward through the network [TML90]:

$$\xi_{is}^j(\tau) = \begin{cases} \delta_{ij}\delta_{s\tau} & \text{if } j \text{ action input unit} \\ 0 & \text{if } \tau{=}1 \wedge j \text{ state/context input unit} \\ \xi_{is}^{j'}(\tau{-}1) & \text{if } \tau{>}1 \wedge j \text{ state/context input unit} \\ & (j' \text{ corresponding output unit of preceding model}) \\ \text{logistic}'(\text{net}_j(\tau)) \cdot \displaystyle\sum_{l\in\text{pred}(j)} \text{weight}_{jl}\, \xi_{is}^l(\tau) & \text{otherwise} \end{cases} \tag{2}$$

The *reinforcement energy* to be minimized is defined as

$$E_{\text{reinf}} \equiv \tfrac{1}{2} \sum_{\tau=1}^{N} \sum_{k} g_k(\tau) \cdot \left(\text{reinf}'_k - \text{activation}_k(\tau)\right)^2 . \tag{3}$$

($k$ numbers the reinforcement output units, $\text{reinf}'_k$ is the desired reinforcement value, usually $\forall k$: $\text{reinf}'_k \equiv 1$, and $g_k$ weights the reinforcement with respect to $\tau$ and $k$, in the simplest case $g_k(\tau) \equiv 1$.) Since $E_{\text{reinf}}$ is differentiable, we can compute the gradient of $E_{\text{reinf}}$ with respect to each particular reinforcement prediction. From these gradients and the gradients $\xi_{is}^k$ of the reinforcement prediction units the gradients

$$\zeta_{is} \equiv \frac{\partial E_{\text{reinf}}}{\partial \, \text{action}_i(s)} = - \sum_{\tau=s}^{N} \sum_{k} g_k(\tau) \cdot \left(\text{reinf}'_k - \text{activation}_k(\tau)\right) \cdot \xi_{is}^k(\tau) \tag{4}$$

are derived which indicate how to change the plan in order to minimize $E_{\text{reinf}}$.

**Variable plan lengths:** The feed-forward manner of the propagation allows it to vary the number of look-ahead steps due to the current accuracy of the model network. Intuitively, if a model network has a relatively large error, looking far into future makes little sense. A good heuristic is to avoid further look-ahead if the current linear error (due to the training patterns) of the model network is larger than the effect of the first action of the plan to the current predictions. This effect is exactly the gradients $\xi_{i1}^k(\tau)$. Using variable plan lengths might overcome the difficulties in finding an appropriate plan length $N$ a priori.

## 2.3   INITIAL PLANS – THE EXPERIENCE NETWORK

It remains to show how to obtain initial plans. There are several basic strategies which are more or less problem-dependent, e.g. random, average over previous actions etc. Obviously, if some planning took place before, the problem of finding an initial plan reduces to the problem of finding a simple action, since the rest of the previous plan is a good candidate for the next initial plan.

A good way of finding this action is the *experience network*. This network is trained to predict the result of the planning procedure by observing the world's state and, in the case of recurrent networks, the temporal context information from the model network. The target values are the results of the planning procedure. Although the experience network is trained like a controller [Bar89], it is used in a different way, since outcoming actions are further optimized by the planning procedure. Thus, even if the knowledge of the experience network lags behind the model network's, the derived actions are optimized with respect to the "knowledge" of the model network rather than the experience network. On the other hand, while the optimization is gradually shifted into the experience network, planning can be progressively shortened.

## 3   APPROACHING A ROLLING BALL WITH A ROBOT ARM

We applied planning with an adaptive world model to a simulation of a real-time robotics task: A robot arm in 3-dimensional space was to approach a rolling ball. Both hand position (i.e. $x, y, z$ and hand angle) and ball position (i.e. $x', y'$) were observed by a camera system in workspace. Conversely, actions were defined as angular changes of the robot joints in configuration space. Model and experience

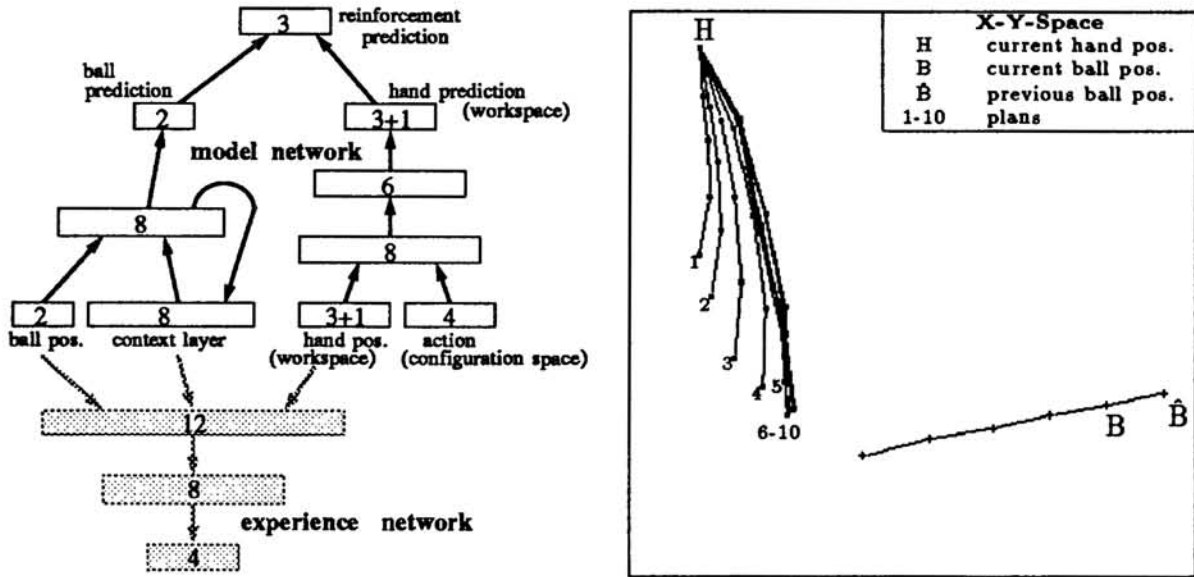

Figure 3: (a) The recurrent model network (white) and the experience network (grey) at the robotics task. (b) Planning: Starting with the initial plan 1, the approximation leads finally to plan 10. The first action of this plan is then performed on the world.

networks are shown in Fig. 3a. Note that the ball movement was predicted by a recurrent Elman-type network, since only the current ball position was visible at any time. The arm prediction is mathematically more sophisticated, because kinematics and inverse kinematics are required to solve it analytically.

The reason why planning makes sense at this task is that we did not want the robot arm to minimize the distance between hand and ball at each step – this would obviously yield trajectories in which the hand follows the ball, e.g.:

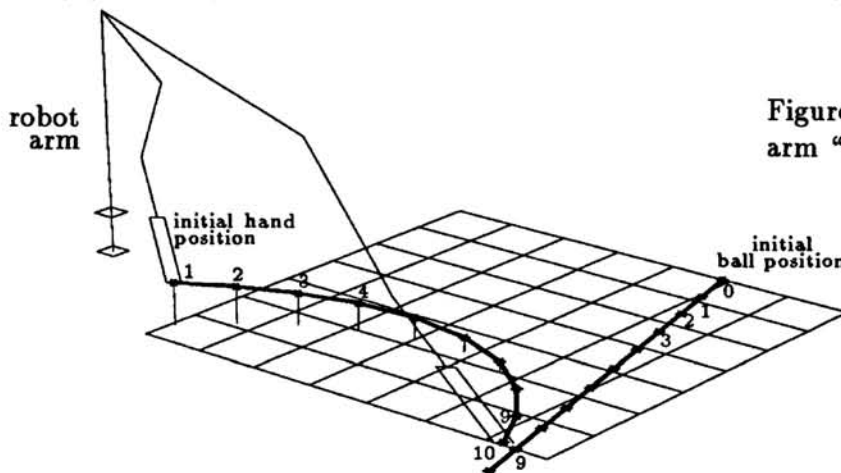

Figure 4: Basic strategy, the arm "follows" the ball.

Instead, we wanted the system to find short cuts by making predictions about the ball's next movement. Thus, the reinforcement measured the distance in workspace. Fig. 3b illustrates a "typical" planning process with look-ahead $N{=}4$, 9 iterations, $g_k(\tau) = 1.3^\tau$ (c.f. (2))$^2$, a weighted stepsize $\eta = 0.05 \cdot 0.9^\tau$, and well-trained model and experience networks. Starting with an initial plan 1 by the experience network

---

$^2$This exponential function is crucial for minimizing later distances rather than the sooner.

the optimization led to plan 10. It is clear to see that the resulting action surpassed the initial plan, which demonstrates the appropriateness of the optimization. The final trajectory was:

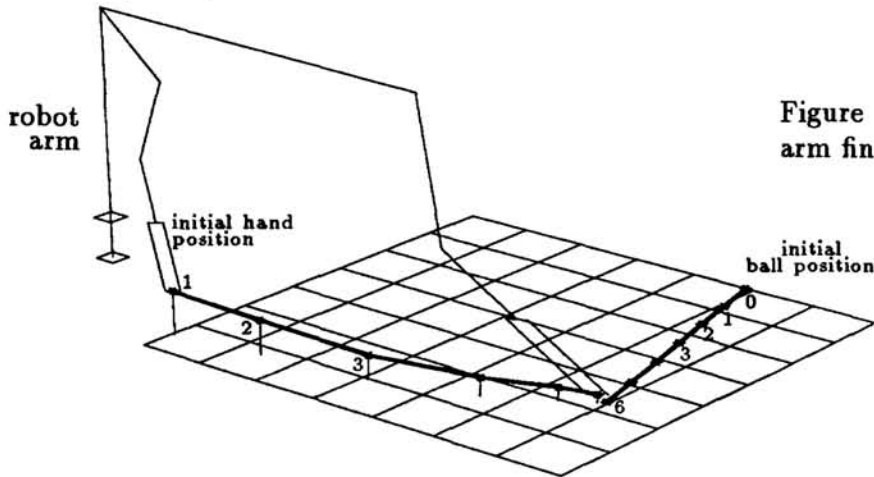

Figure 5: Planning: The arm finds the short cut.

We were now interested in modifying the behavior of the arm. Without further learning of either the model or the experience network, we wanted the arm to approach the ball from above. For this purpose we changed the energy function (7): Before the arm was to approach the ball, the energy was minimal if the arm reached a position exactly above the ball. Since the experience network was not trained for that task, we doubled the number of iteration steps. This led to:

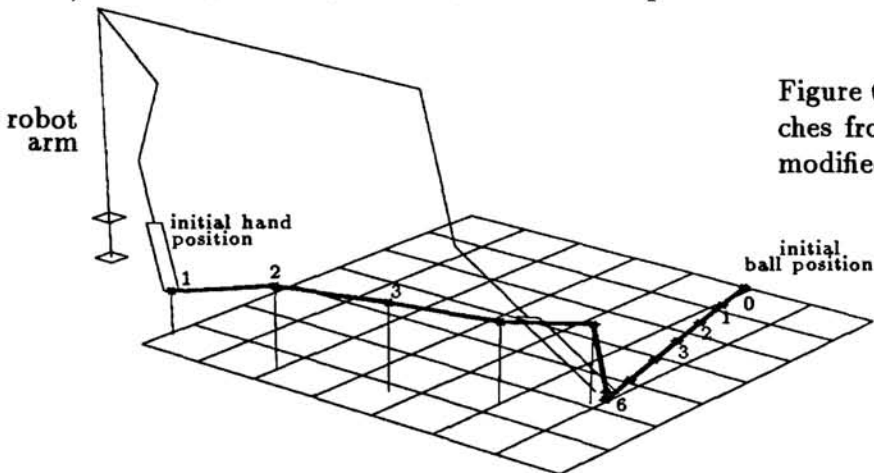

Figure 6: The arm approaches from above due to a modified energy function.

A first implementation on a real robot arm with a camera system showed similar results.

## 4  POLE BALANCING

Next, we applied our planning method to the pole balancing task adopted from [And89]. One main difference to the task described above is the fact that gradient descent is not applicable with binary reinforcement, since the better the approximation by the world model, the more the gradient vanishes. This effect can be prevented by using a second model network with weight decay, which is trained with the same training patterns. Weight decay smoothes the binary mapping. By using the model network for prediction only and the smoothed network for gradient propagation, the pole balancing problem became solvable. We see this as a general

technique for applying gradient descent to binary reinforcement tasks.

We were especially interested in the dependency of look-ahead and the duration of balance. It turned out that in most randomly chosen initial configurations of pole and cart the look-ahead $N = 4$ was sufficient to balance the pole more than 20 000 steps. If the cart is moved randomly, after on average 10 movements the pole falls.

## 5  DISCUSSION

The planning procedure presented in this paper has two crucial limitations. By using a bounded look-ahead, effects of actions to reinforcement beyond this bound can not be taken into account. Even if the plan lengths are kept variable (as described above), each particular planning process must use a finite plan. Moreover, using gradient descent as search heuristic implies the danger of getting stuck in local minima. It might be interesting to investigate other search heuristics.

On the other hand this planning algorithm overcomes certain problems of adaptive controller networks, namely: a) The training is relatively fast, since the model network does not include temporal effects. b) Decisions are optimized due to the current "knowledge" in the system, and no controller lags behind the model network. c) The incorporation of additional constraints to the objective function at runtime is possible, as demonstrated. d) By using a probabilistic experience network the planning algorithm is able to act as a non-deterministic many-to-many controller. Anyway, we have not investigated the latter point yet.

### Acknowledgements

The authors thank Jörg Kindermann and Frank Śmieja for many fruitful discussions and Michael Contzen and Michael Faßbender for their help with the robot arm.

## Footnotes

[1]If an unknown world is to be explored, this action might be disturbed by adding a small random variable.

### References

[And89]  C.W. Anderson. Learning to control an inverted pendulum using neural networks. *IEEE Control Systems Magazine*, 9(3):31–37, 1989.

[Bar89]  A. G. Barto. Connectionist learning for control: An overview. Technical Report COINS TR 89-89, Dept. of Computer and Information Science, University of Massachusetts, Amherst, MA, September 1989.

[Jor89]  M. I. Jordan. Generic constraints on unspecified target constraints. In *Proceedings of the First International Joint Conference on Neural Networks, Washington, DC*, San Diego, 1989. IEEE TAB NN Committee.

[Mun87]  P. Munro. A dual backpropagation scheme for scalar-reward learning. In *Ninth Annual Conference of the Cognitive Science Society*, pages 165–176, Hillsdale, NJ, 1987. Cognitive Science Society, Lawrence Erlbaum.

[Sut84]  R. S. Sutton. *Temporal Credit Assignment in Reinforcement Learning*. PhD thesis, University of Massachusetts, 1984.

[TML90]  S. Thrun, K. Möller, and A. Linden. Adaptive look-ahead planning. In G. Dorffner, editor, *Proceedings KONNAI/OEGAI*, Springer, Sept. 1990.

[TS90]  S. Thrun and F. Śmieja. A general feed-forward algorithm for gradient-descent in connectionist networks. TR 483, GMD, FRG, Nov. 1990.

[WZ88]  R. J. Williams and D. Zipser. A learning algorithm for continually running fully recurrent neural networks. TR ICS Report 8805, Institute for Cognitive Science, University of California, San Diego, CA, 1988.
